# Learning from Multiple Sources

**Koby Crammer, Michael Kearns, Jennifer Wortman**
Department of Computer and Information Science
University of Pennsylvania
Philadelphia, PA 19104

## Abstract

We consider the problem of learning accurate models from multiple sources of "nearby" data. Given distinct samples from multiple data sources and estimates of the dissimilarities between these sources, we provide a general theory of which samples should be used to learn models for each source. This theory is applicable in a broad decision-theoretic learning framework, and yields results for classification and regression generally, and for density estimation within the exponential family. A key component of our approach is the development of approximate triangle inequalities for expected loss, which may be of independent interest.

## 1 Introduction

We introduce and analyze a theoretical model for the problem of learning from multiple sources of "nearby" data. As a hypothetical example of where such problems might arise, consider the following scenario: For each web user in a large population, we wish to learn a classifier for what sites that user is likely to find "interesting." Assuming we have at least a small amount of labeled data for each user (as might be obtained either through direct feedback, or via indirect means such as click-throughs following a search), one approach would be to apply standard learning algorithms to each user's data in isolation. However, if there are natural and accessible measures of similarity between the interests of pairs of users (as might be obtained through their mutual labelings of common web sites), an appealing alternative is to *aggregate* the data of "nearby" users when learning a classifier for each particular user. This alternative is intuitively subject to a trade-off between the increased sample size and how different the aggregated users are.

We treat this problem in some generality and provide a bound addressing the aforementioned trade-off. In our model there are $K$ unknown data sources, with source $i$ generating a distinct sample $S_i$ of $n_i$ observations. We assume we are given only the samples $S_i$, and a *disparity*[1] matrix $D$ whose entry $D(i, j)$ bounds the difference between source $i$ and source $j$. Given these inputs, we wish to decide which subset of the samples $S_j$ will result in the best model for each source $i$. Our framework includes settings in which the sources produce data for classification, regression, and density estimation (and more generally any additive-loss learning problem obeying certain conditions).

Our main result is a general theorem establishing a bound on the expected loss incurred by using all data sources within a given disparity of the target source. Optimization of this bound then yields a recommended subset of the data to be used in learning a model of each source. Our bound clearly expresses a trade-off between three quantities: the sample size used (which increases as we include data from more distant models), a weighted average of the disparities of the sources whose data is used, and a model complexity term. It can be applied to any learning setting in which the underlying loss function obeys an *approximate* triangle inequality, and in which the class of hypothesis models under consideration obeys uniform convergence of empirical estimates of loss to expectations.

For classification problems, the standard triangle inequality holds. For regression we prove a 2-approximation to the triangle inequality, and for density estimation for members of the exponential family, we apply Bregman divergence techniques to provide approximate triangle inequalities. We believe these approximations may find independent applications within machine learning. Uniform convergence bounds for the settings we consider may be obtained via standard data-independent model complexity measures such as VC dimension and pseudo-dimension, or via more recent data-dependent approaches such as Rademacher complexity.

The research described here grew out of an earlier paper by the same authors [1] which examined the considerably more limited problem of learning a model when all data sources are corrupted versions of a *single, fixed* source, for instance when each data source provides noisy samples of a fixed binary function, but with varying levels of noise. In the current work, each source may be entirely unrelated to all others except as constrained by the bounds on disparities, requiring us to develop new techniques. Wu and Dietterich studied similar problems experimentally in the context of SVMs [2]. The framework examined here can also be viewed as a type of transfer learning [3, 4].

In Section 2 we introduce a decision-theoretic framework for probabilistic learning that includes classification, regression, density estimation and many other settings as special cases, and then give our multiple source generalization of this model. In Section 3 we provide our main result, which is a general bound on the expected loss incurred by using all data within a given disparity of a target source. Section 4 then applies this bound to a variety of specific learning problems. In Section 5 we briefly examine data-dependent applications of our general theory using Rademacher complexity.

## 2    Learning models

Before detailing our multiple-source learning model, we first introduce a standard decision-theoretic learning framework in which our goal is to find a model minimizing a generalized notion of empirical loss [5]. Let the *hypothesis class* $\mathcal{H}$ be a set of models (which might be classifiers, real-valued functions, densities, etc.), and let $f$ be the *target model*, which may or may not lie in the class $\mathcal{H}$. Let $z$ be a (generalized) data point or observation. For instance, in (noise-free) classification and regression, $z$ will consist of a pair $\langle x, y \rangle$ where $y = f(x)$. In density estimation, $z$ is the observed value $x$. We assume that the target model $f$ induces some underlying distribution $P_f$ over observations $z$. In the case of classification or regression, $P_f$ is induced by drawing the inputs $x$ according to some underlying distribution $P$, and then setting $y = f(x)$ (possibly corrupted by noise). In the case of density estimation $f$ simply defines a distribution $P_f$ over observations $x$.

Each setting we consider has an associated *loss function* $\mathcal{L}(h, z)$. For example, in classification we typically consider the 0/1 loss: $\mathcal{L}(h, \langle x, y \rangle) = 0$ if $h(x) = y$, and 1 otherwise. In regression we might consider the squared loss function $\mathcal{L}(h, \langle x, y \rangle) = (y - h(x))^2$. In density estimation we might consider the log loss $\mathcal{L}(h, x) = \log(1/h(x))$. In each case, we are interested in the expected loss of a model $g_2$ on target $g_1$, $e(g_1, g_2) = \mathrm{E}_{z \sim P_{g_1}}[\mathcal{L}(g_2, z)]$. Expected loss is not necessarily symmetric.

In our multiple source model, we are presented with $K$ distinct samples or *piles* of data $S_1, ..., S_K$, and a symmetric $K \times K$ matrix $D$. Each pile $S_i$ contains $n_i$ observations that are generated from a fixed and unknown model $f_i$, and $D$ satisfies $e(f_i, f_j), e(f_j, f_i) \leq D(i, j)$. [2] Our goal is to decide which piles $S_j$ to use in order to learn the best approximation (in terms of expected loss) to each $f_i$.

While we are interested in accomplishing this goal for each $f_i$, it suffices and is convenient to examine the problem from the perspective of a fixed $f_i$. Thus without loss of generality let us suppose that we are given piles $S_1, ..., S_K$ of size $n_1, \ldots, n_K$ from models $f_1, \ldots, f_K$ such that $\epsilon_1 \equiv D(1, 1) \leq \epsilon_2 \equiv D(1, 2) \leq \cdots \leq \epsilon_K \equiv D(1, K)$, and our goal is to learn $f_1$. Here we have simply taken the problem in the preceding paragraph, focused on the problem for $f_1$, and reordered the other models according to their proximity to $f_1$. To highlight the distinguished role of the target $f_1$ we shall denote it $f$. We denote the observations in $S_j$ by $z_1^j, \ldots, z_{n_j}^j$. In all cases we will analyze, for any $k \leq K$, the hypothesis $\hat{h}_k$ minimizing the empirical loss $\hat{e}_k(h)$ on the first $k$ piles $S_1, \ldots, S_k$, i.e.

$$\hat{h}_k = \operatorname*{argmin}_{h \in \mathcal{H}} \hat{e}_k(h) = \operatorname*{argmin}_{h \in \mathcal{H}} \frac{1}{n_{1:k}} \sum_{j=1}^{k} \sum_{i=1}^{n_j} \mathcal{L}(h, z_i^j)$$

where $n_{1:k} = n_1 + \cdots + n_k$. We also denote the expected error of function $h$ with respect to the first $k$ piles of data as

$$e_k(h) = \mathrm{E}\left[\hat{e}_k(h)\right] = \sum_{i=1}^{k} \left(\frac{n_i}{n_{1:k}}\right) e(f_i, h).$$

## 3  General theory

In this section we provide the first of our main results: a general bound on the expected loss of the model minimizing the empirical loss on the nearest $k$ piles. Optimization of this bound leads to a recommended number of piles to incorporate when learning $f = f_1$. The key ingredients needed to apply this bound are an approximate triangle inequality and a uniform convergence bound, which we define below. In the subsequent sections we demonstrate that these ingredients can indeed be provided for a variety of natural learning problems.

**Definition 1** *For $\alpha \geq 1$, we say that the $\alpha$-**triangle inequality** holds for a class of models $\mathcal{F}$ and expected loss function $e$ if for all $g_1, g_2, g_3 \in \mathcal{F}$ we have*

$$e(g_1, g_2) \leq \alpha(e(g_1, g_3) + e(g_3, g_2)).$$

*The parameter $\alpha \geq 1$ is a constant that depends on $\mathcal{F}$ and $e$.*

The choice $\alpha = 1$ yields the standard triangle inequality. We note that the restriction to models in the class $\mathcal{F}$ may in some cases be quite weak — for instance, when $\mathcal{F}$ is all possible classifiers or real-valued functions with bounded range — or stronger, as in densities from the exponential family. Our results will require only that the unknown *source* models $f_1, \ldots, f_K$ lie in $\mathcal{F}$, even when our *hypothesis* models are chosen from some possibly much more restricted class $\mathcal{H} \subseteq \mathcal{F}$. For now we simply leave $\mathcal{F}$ as a parameter of the definition.

**Definition 2** *A **uniform convergence bound** for a hypothesis space $\mathcal{H}$ and loss function $\mathcal{L}$ is a bound that states that for any $0 < \delta < 1$, with probability at least $1 - \delta$ for any $h \in \mathcal{H}$*

$$|\hat{e}(h) - e(h)| \leq \beta(n, \delta)$$

*where $\hat{e}(h) = \frac{1}{n} \sum_{i=1}^{n} \mathcal{L}(h, z_i)$ for $n$ observations $z_1, \ldots, z_n$ generated independently according to distributions $P_1, \ldots P_n$, and $e(h) = \mathrm{E}\left[\hat{e}(h)\right]$ where the expectation is taken over $z_1, \ldots, z_n$. $\beta$ is a function of the number of observations $n$ and the confidence $\delta$, and depends on $\mathcal{H}$ and $\mathcal{L}$.*

This definition simply asserts that for every model in $\mathcal{H}$, its empirical loss on a sample of size $n$ and the expectation of this loss will be "close." In general the function $\beta$ will incorporate standard measures of the complexity of $\mathcal{H}$, and will be a decreasing function of the sample size $n$, as in the classical $\mathcal{O}(\sqrt{d/n})$ bounds of VC theory. Our bounds will be derived from the rich literature on uniform convergence. The only twist to our setting is the fact that the observations are no longer necessarily identically distributed, since they are generated from multiple sources. However, generalizing the standard uniform convergence results to this setting is straightforward.

We are now ready to present our general bound.

**Theorem 1** *Let $e$ be the expected loss function for loss $\mathcal{L}$, and let $\mathcal{F}$ be a class of models for which the $\alpha$-triangle inequality holds with respect to $e$. Let $\mathcal{H} \subseteq \mathcal{F}$ be a class of hypothesis models for which there is a uniform convergence bound $\beta$ for $\mathcal{L}$. Let $K \in \mathbb{N}$, $f = f_1, f_2, \ldots, f_K \in \mathcal{F}$, $\{\epsilon_i\}_{i=1}^{K}$, $\{n_i\}_{i=1}^{K}$, and $\hat{h}_k$ be as defined above. For any $\delta$ such that $0 < \delta < 1$, with probability at least $1 - \delta$, for any $k \in \{1, \ldots, K\}$*

$$e(f, \hat{h}_k) \leq (\alpha + \alpha^2) \sum_{i=1}^{k} \left(\frac{n_i}{n_{1:k}}\right) \epsilon_i + 2\alpha\beta(n_{1:k}, \delta/2K) + \alpha^2 \min_{h \in \mathcal{H}} \{e(f, h)\}$$

Before providing the proof, let us examine the bound of Theorem 1, which expresses a natural and intuitive trade-off. The first term in the bound is a weighted sum of the disparities of the $k \leq K$ models whose data is used with respect to the target model $f = f_1$. We expect this term to *increase* as we increase $k$ to include more distant piles. The second term is determined by the uniform convergence bound. We expect this term to *decrease* with added piles due to the increased sample size. The final term is what is typically called the *approximation error* — the residual loss that we incur simply by limiting our hypothesis model to fall in the restricted class $\mathcal{H}$. All three terms are influenced by the strength of the approximate triangle inequality that we have, as quantified by $\alpha$.

The bounds given in Theorem 1 can be loose, but provide an upper bound necessary for optimization and suggest a natural choice for the number of piles $k^*$ to use to estimate the target $f$:

$$k^* = \operatorname*{argmin}_{k} \left( (\alpha + \alpha^2) \sum_{i=1}^{k} \left( \frac{n_i}{n_{1:k}} \right) \epsilon_i + 2\alpha\beta(n_{1:k}, \delta/2K) \right).$$

Theorem 1 and this optimization make the implicit assumption that the best subset of piles to use will be a prefix of the piles — that is, that we should not "skip" a nearby pile in favor of more distant ones. This assumption will generally be true for typical data-independent uniform convergence such as VC dimension bounds, and true on average for data-dependent bounds, where we expect uniform convergence bounds to improve with increased sample size. We now give the proof of Theorem 1.

**Proof:** (Theorem 1) By Definition 1, for any $h \in \mathcal{H}$, any $k \in \{1, \ldots K\}$, and any $i \in \{1, \ldots, k\}$,

$$\left( \frac{n_i}{n_{1:k}} \right) e(f, h) \leq \left( \frac{n_i}{n_{1:k}} \right) (\alpha e(f, f_i) + \alpha e(f_i, h))$$

Summing over all $i \in \{1, \ldots, k\}$, we find

$$
\begin{aligned}
e(f, h) &\leq \sum_{i=1}^{k} \left( \frac{n_i}{n_{1:k}} \right) (\alpha e(f, f_i) + \alpha e(f_i, h)) \\
&= \alpha \sum_{i=1}^{k} \left( \frac{n_i}{n_{1:k}} \right) e(f, f_i) + \alpha \sum_{i=1}^{k} \left( \frac{n_i}{n_{1:k}} \right) e(f_i, h) \leq \alpha \sum_{i=1}^{k} \left( \frac{n_i}{n_{1:k}} \right) \epsilon_i + \alpha e_k(h)
\end{aligned}
$$

In the first line above we have used the $\alpha$-triangle inequality to deliberately introduce a weighted summation involving the $f_i$. In the second line, we have broken up the summation. Notice that the first summation is a weighted average of the expected loss of each $f_i$, while the second summation is the expected loss of $h$ on the data. Using the uniform convergence bound, we may assert that with high probability $e_k(h) \leq \hat{e}_k(h) + \beta(n_{1:k}, \delta/2K)$, and with high probability

$$\hat{e}_k(\hat{h}_k) = \min_{h \in \mathcal{H}}\{\hat{e}_k(h)\} \leq \min_{h \in \mathcal{H}} \left\{ \sum_{i=1}^{k} \left( \frac{n_i}{n_{1:k}} \right) e(f_i, h) + \beta(n_{1:k}, \delta/2K) \right\}$$

Putting these pieces together, we find that with high probability

$$
\begin{aligned}
e(f, \hat{h}_k) &\leq \alpha \sum_{i=1}^{k} \left( \frac{n_i}{n_{1:k}} \right) \epsilon_i + 2\alpha\beta(n_{1:k}, \delta/2K) + \alpha \min_{h \in \mathcal{H}} \left\{ \sum_{i=1}^{k} \left( \frac{n_i}{n_{1:k}} \right) e(f_i, h) \right\} \\
&\leq \alpha \sum_{i=1}^{k} \left( \frac{n_i}{n_{1:k}} \right) \epsilon_i + 2\alpha\beta(n_{1:k}, \delta/2K) \\
&\quad + \alpha \min_{h \in \mathcal{H}} \left\{ \sum_{i=1}^{k} \left( \frac{n_i}{n_{1:k}} \right) \alpha e(f_i, f) + \sum_{i=1}^{k} \left( \frac{n_i}{n_{1:k}} \right) \alpha e(f, h) \right\} \\
&= (\alpha + \alpha^2) \sum_{i=1}^{k} \left( \frac{n_i}{n_{1:k}} \right) \epsilon_i + 2\alpha\beta(n_{1:k}, \delta/2K) + \alpha^2 \min_{h \in \mathcal{H}} \{e(f, h)\}
\end{aligned}
$$

∎

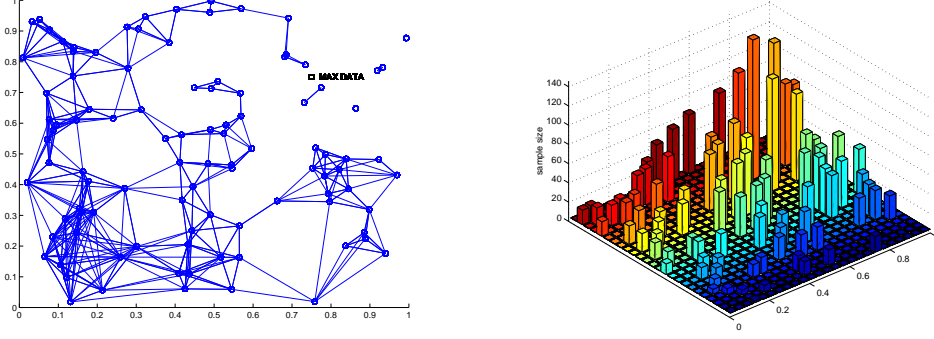

Figure 1: Visual demonstration of Theorem 2. In this problem there are $K = 100$ classifiers, each defined by 2 parameters represented by a point $f_i$ in the unit square, such that the expected disagreement rate between two such classifiers equals the $L_1$ distance between their parameters. (It is easy to create simple input distributions and classifiers that generate exactly this geometry.) We chose the 100 parameter vectors $f_i$ uniformly at random from the unit square (the circles in the left panel). To generate varying pile sizes, we let $n_i$ decrease with the distance of $f_i$ from a chosen "central" point at $(0.75, 0.75)$ (marked "MAX DATA" in the left panel); the resulting pile sizes for each model are shown in the bar plot in the right panel, where the origin $(0, 0)$ is in the near corner, $(1, 1)$ in the far corner, and the pile sizes clearly peak near $(0.75, 0.75)$. Given these $f_i$, $n_i$ and the pairwise distances, the undirected graph on the left includes an edge between $f_i$ and $f_j$ if and only if the data from $f_j$ is used to learn $f_i$ and/or the converse when Theorem 2 is used to optimize the distance of the data used. The graph simultaneously displays the geometry implicit in Theorem 2 as well as its adaptivity to local circumstances. Near the central point the graph is quite sparse and the edges quite short, corresponding to the fact that for such models we have enough direct data that it is not advantageous to include data from distant models. Far from the central point the graph becomes dense and the edges long, as we are required to aggregate a larger neighborhood to learn the optimal model. In addition, decisions are affected locally by how many models are "nearby" a given model.

## 4 Applications to standard learning settings

In this section we demonstrate the applicability of the general theory given by Theorem 1 to several standard learning settings. We begin with the most straightforward application, classification.

### 4.1 Binary classification

In binary classification, we assume that our target model is a fixed, unknown and arbitrary function $f$ from some input set $\mathcal{X}$ to $\{0, 1\}$, and that there is a fixed and unknown distribution $P$ over the $\mathcal{X}$. Note that the distribution $P$ over input does not depend on the target function $f$. The observations are of the form $z = \langle x, y \rangle$ where $y \in \{0, 1\}$. The loss function $\mathcal{L}(h, \langle x, y \rangle)$ is defined as 0 if $y = h(x)$ and 1 otherwise, and the corresponding expected loss is $e(g_1, g_2) = \mathrm{E}_{\langle x, y \rangle \sim P_{g_1}} [\mathcal{L}(g_2, \langle x, y \rangle)] = \Pr_{x \sim P}[g_1(x) \neq g_2(x)]$. For 0/1 loss it is well-known and easy to see that the (standard) 1-triangle inequality holds, and classical VC theory [6] provides us with uniform convergence. The conditions of Theorem 1 are thus easily satisfied, yielding the following.

**Theorem 2** *Let $\mathcal{F}$ be the set of all functions from an input set $\mathcal{X}$ into $\{0,1\}$ and let $d$ be the VC dimension of $\mathcal{H} \subseteq \mathcal{F}$. Let $e$ be the expected 0/1 loss. Let $K \in \mathbb{N}$, $f = f_1, f_2, \ldots, f_K \in \mathcal{F}$, $\{\epsilon_i\}_{i=1}^K$, $\{n_i\}_{i=1}^K$, and $\hat{h}_k$ be as defined above in the multi-source learning model. For any $\delta$ such that $0 < \delta < 1$, with probability at least $1 - \delta$, for any $k \in \{1, \ldots, K\}$*

$$e(f, \hat{h}_k) \leq 2 \sum_{i=1}^k \left( \frac{n_i}{n_{1:k}} \right) \epsilon_i + \min_{h \in \mathcal{H}} \{e(f, h)\} + 2\sqrt{\frac{d \log \left( 2e n_{1:k}/d \right) + \log \left( 16K/\delta \right)}{8 n_{1:k}}}$$

In Figure 1 we provide a visual demonstration of the behavior of Theorem 1 applied to a simple classification problem.

## 4.2 Regression

We now turn to regression with squared loss. Here our target model $f$ is any function from an input class $\mathcal{X}$ into some bounded subset of $\mathbb{R}$. (Frequently we will have $\mathcal{X} \subseteq \mathbb{R}^d$, but this is not required.) We again assume a fixed but unknown distribution $P$ (that does not depend on $f$) on the inputs. Our observations are of the form $z = \langle x, y \rangle$. Our loss function is $\mathcal{L}(h, \langle x, y \rangle) = (y - h(x))^2$, and the expected loss is thus $e(g_1, g_2) = \mathrm{E}_{\langle x,y \rangle \sim P_{g_1}} [\mathcal{L}(g_2, \langle x, y \rangle)] = \mathrm{E}_{x \sim P} [(g_1(x) - g_2(x))^2]$.

For regression it is known that the standard 1-triangle inequality does not hold. However, a 2-triangle inequality does hold and is stated in the following lemma. The proof is given in Appendix A. [3]

**Lemma 1** *Given any three functions $g_1, g_2, g_3 : \mathcal{X} \to \mathbb{R}$, a fixed and unknown distribution $P$ on the inputs $\mathcal{X}$, and the expected loss $e(g_1, g_2) = \mathrm{E}_{x \sim P} [(g_1(x) - g_2(x))^2]$,*

$$e(g_1, g_2) \leq 2 (e(g_1, g_3) + e(g_3, g_1)).$$

The other required ingredient is a uniform convergence bound for regression with squared loss. There is a rich literature on such bounds and their corresponding complexity measures for the model class $\mathcal{H}$, including the fat-shattering generalization of VC dimension [7], $\epsilon$-nets and entropy [6] and the combinatorial and pseudo-dimension approaches beautifully surveyed in [5]. For concreteness here we adopt the latter approach, since it serves well in the following section on density estimation.

While a detailed exposition of the pseudo-dimension $\dim(\mathcal{H})$ of a class $\mathcal{H}$ of real-valued functions exceeds both our space limitations and scope, it suffices to say that it generalizes the VC dimension for binary functions and plays a similar role in uniform convergence bounds. More precisely, in the same way that the VC dimension measures the largest set of points on which a set of classifiers can exhibit "arbitrary" behavior (by achieving all possible labelings of the points), $\dim(\mathcal{H})$ measures the largest set of points on which the output values induced by $\mathcal{H}$ are "full" or "space-filling." (Technically we ask whether $\{\langle h(x_1), \ldots, h(x_d) \rangle : h \in \mathcal{H}\}$ intersects all orthants of $\mathbb{R}^d$ with respect to some chosen origin.) Ignoring constant and logarithmic factors, uniform convergence bounds can be derived in which the complexity penalty is $\sqrt{\dim(\mathcal{H})/n}$. As with the VC dimension, $\dim(\mathcal{H})$ is ordinarily closely related to the number of free parameters defining $\mathcal{H}$. Thus for linear functions in $\mathbb{R}^d$ it is $\mathcal{O}(d)$ and for neural networks with $W$ weights it is $\mathcal{O}(W)$, and so on.

Careful application of pseudo-dimension results from [5] along with Lemma 1 and Theorem 1 yields the following. A sketch of the proof appears in Appendix A.

**Theorem 3** *Let $\mathcal{F}$ be the set of functions from $\mathcal{X}$ into $[-B, B]$ and let $d$ be the pseudo-dimension of $\mathcal{H} \subseteq \mathcal{F}$ under squared loss. Let $e$ be the expected squared loss. Let $K \in \mathbb{N}$, $f = f_1, f_2, \ldots, f_K \in \mathcal{F}$, $\{\epsilon_i\}_{i=1}^K$, $\{n_i\}_{i=1}^K$, and $\hat{h}_k$ be as defined in the multi-source learning model. Assume that $n_1 \geq d/16e$. For any $\delta$ such that $0 < \delta < 1$, with probability at least $1 - \delta$, for any $k \in \{1, \ldots, K\}$*

$$e(f, \hat{h}_k) \leq 6 \sum_{i=1}^{k} \left( \frac{n_i}{n_{1:k}} \right) \epsilon_i + 4 \min_{h \in \mathcal{H}} \{e(f, h)\} + 128 B^2 \left( \sqrt{\frac{d}{n_{1:k}}} + \sqrt{\frac{\ln(16K/\delta)}{n_{1:k}}} \right) \left( \sqrt{\ln \frac{16 e^2 n_{1:k}}{d}} \right)$$

## 4.3 Density estimation

We turn to the more complex application to density estimation. Here our models are no longer functions, but densities $P$. The loss function for an observation $x$ is the log loss $\mathcal{L}(P, x) = \log (1/P(x))$. The expected loss is then $e(P_1, P_2) = \mathrm{E}_{x \sim P_1} [\mathcal{L}(P_2, x)] = \mathrm{E}_{x \sim P_1} [\log(1/P_2(x))]$.

As we are not aware of an $\alpha$-triangle inequality that holds simultaneously for all density functions, we provide general mathematical tools to derive specialized $\alpha$-triangle inequalities for specific classes of distributions. We focus on the exponential family of distributions, which is quite general and has nice properties which allow us to derive the necessary machinery to apply Theorem 1. We start by defining the exponential family and explaining some of its properties. We proceed by deriving an $\alpha$-triangle inequality for Kullback-Liebler divergence in exponential families that implies

an $\alpha$-triangle inequality for our expected loss function. This inequality and a uniform convergence bound based on pseudo-dimension yield a general method for deriving error bounds in the multiple source setting which we illustrate using the example of multinomial distributions.

Let $x \in \mathcal{X}$ be a random variable, in either a continuous space (e.g. $\mathcal{X} \subseteq \mathbb{R}^d$) or a discrete space (e.g. $\mathcal{X} \subseteq \mathbb{Z}^d$). We define the exponential family of distributions in terms of the following components. First, we have a vector function of the *sufficient statistics* needed to compute the distribution, denoted $\Psi : \mathbb{R}^d \to \mathbb{R}^{d'}$. Associated with $\Psi$ is a vector of *expectation parameters* $\mu \in \mathbb{R}^{d'}$ which parameterizes a particular distribution. Next we have a convex vector function $F : \mathbb{R}^{d'} \to \mathbb{R}$ (defined below) which is unique for each family of exponential distributions, and a normalization function $P_0(x)$. Using this notation we define a probability distribution (in the expectation parameters) to be

$$\mathrm{P}_F\left(x \mid \mu\right) = e^{\nabla F(\mu) \cdot (\Psi(x) - \mu) + F(\mu)} P_0(x) . \tag{1}$$

For all distributions we consider it will hold that $\mathrm{E}_{x \sim \mathrm{P}_F(\cdot \mid \mu)}\left[\Psi(x)\right] = \mu$. Using this fact and the linearity of expectation, we can derive the Kullback-Liebler (KL) divergence between two distributions of the same family (which use the same functions $F$ and $\Psi$) and obtain

$$\mathrm{KL}\left(\mathrm{P}_F\left(x \mid \mu_1\right) \parallel \mathrm{P}_F\left(x \mid \mu_2\right)\right) \quad = \quad F(\mu_1) - \left[F(\mu_2) + \nabla F(\mu_2) \cdot (\mu_1 - \mu_2)\right] . \tag{2}$$

We define the quantity on the right to be the *Bregman divergence* between the two (parameter) vectors $\mu_1$ and $\mu_2$, denoted $\mathrm{B}_F\left(\mu_1 \parallel \mu_2\right)$. The Bregman divergence measures the difference between $F$ and its first-order Taylor expansion about $\mu_2$ evaluated at $\mu_1$. Eq. (2) states that the KL divergence between two members of the exponential family is equal to the Bregman divergence between the two corresponding expectation parameters. We refer the reader to [8] for more details about Bregman divergences and to [9] for more information about exponential families.

We will use the above relation between the KL divergence for exponential families and Bregman divergences to derive a triangle inequality as required by our theory. The following lemma shows that if we can provide a triangle inequality for the KL function, we can do so for expected log loss.

**Lemma 2** *Let $e$ be the expected log loss, i.e. $e(P_1, P_2) = \mathrm{E}_{x \sim P_1}\left[\log(1/P_2(x))\right]$. For any three probability distributions $P_1$, $P_2$, and $P_3$, if $KL\left(P_1 \parallel P_2\right) \leq \alpha(KL\left(P_1 \parallel P_3\right) + KL\left(P_3 \parallel P_2\right))$ for some $\alpha \geq 1$ then $e(P_1, P_2) \leq \alpha(e(P_1, P_3) + e(P_3, P_2))$.*

The proof is given in Appendix B. The next lemma gives an approximate triangle inequality for the KL divergence. We assume that there exists a closed set $\mathcal{P} = \{\mu\}$ which contains all the parameter vectors. The proof (again see Appendix B) uses Taylor's Theorem to derive upper and lower bounds on the Bregman divergence and then uses Eq. (2) to relate these bounds to the KL divergence.

**Lemma 3** *Let $P_1$, $P_2$, and $P_3$ be distributions from an exponential family with parameters $\mu$ and function $F$. Then*

$$KL\left(P_1 \parallel P_2\right) \leq \alpha \left(KL\left(P_1 \parallel P_3\right) + KL\left(P_3 \parallel P_2\right)\right)$$

*where $\alpha = 2 \sup_{\xi \in \mathcal{P}} \lambda_1(H(F(\xi))) / \inf_{\xi \in \mathcal{P}} \lambda_{d'}(H(F(\xi)))$. Here $\lambda_1(\cdot)$ and $\lambda_{d'}(\cdot)$ are the highest and lowest eigenvalues of a given matrix, and $H(\cdot)$ is the Hessian matrix.*

The following theorem, which states bounds for multinomial distributions in the multi-source setting, is provided to illustrate the type of results that can be obtained using the machinery described in this section. More details on the application to the multinomial distribution are given in Appendix B.

**Theorem 4** *Let $\mathcal{F} \equiv \mathcal{H}$ be the set of multinomial distributions over $N$ values with the probability of each value bounded from below by $\gamma$ for some $\gamma > 0$, and let $\alpha = 2/\gamma$. Let $d$ be the pseudo-dimension of $\mathcal{H}$ under log loss, and let $e$ be the expected log loss. Let $K \in \mathbb{N}$, $f = f_1, f_2, \ldots, f_K \in \mathcal{F}$, $\{\epsilon_i\}_{i=1}^K$, [4] $\{n\}_{i=1}^K$, and $\hat{h}_k$ be as defined above in the multi-source learning model. Assume that $n_1 \geq d/16\mathrm{e}$. For any $0 < \delta < 1$, with probability at least $1 - \delta$ for any $k \in \{1, \ldots, K\}$,*

$$e(f, \hat{h}_k) \quad \leq \quad (\alpha + \alpha^2) \sum_{i=1}^k \left(\frac{n_i}{n_{1:k}}\right) \epsilon_i + \alpha \min_{h \in \mathcal{H}} \{e(f, h)\}$$

$$+ 128 \log^2\left(\frac{\alpha}{2}\right)\left(\sqrt{\frac{d}{n_{1:k}}} + \sqrt{\frac{\ln(16K/\delta)}{n_{1:k}}}\right)\left(\sqrt{\ln\frac{16\mathrm{e}^2 n_{1:k}}{d}}\right)$$

## 5   Data-dependent bounds

Given the interest in data-dependent convergence methods (such as maximum margin, PAC-Bayes, and others) in recent years, it is natural to ask how our multi-source theory can exploit these modern bounds. We examine one specific case for classification here using Rademacher complexity [10, 11]; analogs can be derived in a similar manner for other learning problems.

If $\mathcal{H}$ is a class of functions mapping from a set $\mathcal{X}$ to $\mathbb{R}$, we define the *empirical Rademacher complexity* of $\mathcal{H}$ on a fixed set of observations $x_1, \ldots, x_n$ as

$$\hat{R}_n(\mathcal{H}) = \mathrm{E}\left[\sup_{h \in \mathcal{H}} \left|\frac{2}{n}\sum_{i=1}^{n} \sigma_i h(x_i)\right| \,\Big|\, x_1, \ldots, x_n\right]$$

where the expectation is taken over independent uniform $\{\pm 1\}$-valued random variables $\sigma_1, \ldots, \sigma_n$. The Rademacher complexity for $n$ observations is then defined as $R_n(\mathcal{H}) = \mathrm{E}\left[\hat{R}_n(\mathcal{H})\right]$ where the expectation is over $x_1, \ldots, x_n$.

We can apply Rademacher-based convergence bounds to obtain a data-dependent multi-source bound for classification. A proof sketch using techniques and theorems of [10] is in Appendix C.

**Theorem 5** *Let $\mathcal{F}$ be the set of all functions from an input set $\mathcal{X}$ into $\{-1,1\}$ and let $\hat{R}_{n_{1:k}}$ be the empirical Rademacher complexity of $\mathcal{H} \subseteq \mathcal{F}$ on the first $k$ piles of data. Let $e$ be the expected 0/1 loss. Let $K \in \mathbb{N}$, $f = f_1, f_2, \ldots, f_K \in \mathcal{F}$, $\{\epsilon_i\}_{i=1}^{K}$, $\{n_i\}_{i=1}^{K}$, and $\hat{h}_k$ be as defined in the multi-source learning model. Assume that $n_1 \geq d/16\mathrm{e}$. For any $\delta$ such that $0 < \delta < 1$, with probability at least $1 - \delta$, for any $k \in \{1, \ldots, K\}$*

$$e(f, \hat{h}_k) \leq 2\sum_{i=1}^{k}\left(\frac{n_i}{n_{1:k}}\right)\epsilon_i + \min_{h \in \mathcal{H}}\{e(f, h)\} + \hat{R}_{n_{1:k}}(\mathcal{H}) + 4\sqrt{\frac{2\ln(4K/\delta)}{n_{1:k}}}$$

While the use of data-dependent complexity measures can be expected to yield more accurate bounds and thus better decisions about the number $k^*$ of piles to use, it is not without its costs in comparison to the more standard data-independent approaches. In particular, in principle the optimization of the bound of Theorem 5 to choose $k^*$ may actually involve running the learning algorithm on all possible prefixes of the piles, since we cannot know the data-dependent complexity term for each prefix without doing so. In contrast, the data-independent bounds can be computed and optimized for $k^*$ without examining the data at all, and the learning performed only once on the first $k^*$ piles.

## Footnotes

[1] We avoid using the term distance since our results include settings in which the underlying loss measures may not be formal distances.

[2]While it may seem restrictive to assume that $D$ is given, notice that $D(i, j)$ can be often be estimated from data, for example in a classification setting in which common instances labeled by both $f_i$ and $f_j$ are available.

[3] A version of this paper with the appendix included can be found on the authors' websites.

[4]Here we can actually make the weaker assumption that the $\epsilon_i$ bound the KL divergences rather than the expected log loss, which avoids our needing upper bounds on the entropy of each source distribution.

### References

[1]  K. Crammer, M. Kearns, and J. Wortman. Learning from data of variable quality. In *NIPS 18*, 2006.

[2]  P. Wu and T. Dietterich. Improving SVM accuracy by training on auxiliary data sources. In *ICML*, 2004.

[3]  J. Baxter. Learning internal representations. In *COLT*, 1995.

[4]  S. Ben-David. Exploiting task relatedness for multiple task learning. In *COLT*, 2003.

[5]  D. Haussler. Decision theoretic generalizations of the PAC model for neural net and other learning applications. *Information and Computation*, 1992.

[6]  V. N. Vapnik. *Statistical Learning Theory*. Wiley, 1998.

[7]  M. Kearns and R. Schapire. Efficient distribution-free learning of probabilistic concepts. *JCSS*, 1994.

[8]  Y. Censor and S.A. Zenios. *Parallel Optimization: Theory, Algorithms, and Applications*. Oxford University Press, New York, NY, USA, 1997.

[9]  M. J. Wainwright and M. I. Jordan. Graphical models, exponential families, and variational inference. Technical Report 649, Department of Statistics, University of California, Berkeley, 2003.

[10]  P. L. Bartlett and S. Mendelson. Rademacher and Gaussian complexities: Risk bounds and structural results. *Journal of Machine Learning Research*, 2002.

[11]  V. Koltchinskii. Rademacher penalties and structural risk minimization. *IEEE Trans. Info. Theory*, 2001.
